# Online Optimization in $\mathcal{X}$-Armed Bandits

**Sébastien Bubeck**
INRIA Lille, SequeL project, France
`sebastien.bubeck@inria.fr`

**Rémi Munos**
INRIA Lille, SequeL project, France
`remi.munos@inria.fr`

**Gilles Stoltz**
Ecole Normale Supérieure and HEC Paris
`gilles.stoltz@ens.fr`

**Csaba Szepesvári**
Department of Computing Science, University of Alberta
`szepesva@cs.ualberta.ca` [*]

## Abstract

We consider a generalization of stochastic bandit problems where the set of arms, $\mathcal{X}$, is allowed to be a generic topological space. We constraint the mean-payoff function with a dissimilarity function over $\mathcal{X}$ in a way that is more general than Lipschitz. We construct an arm selection policy whose regret improves upon previous result for a large class of problems. In particular, our results imply that if $\mathcal{X}$ is the unit hypercube in a Euclidean space and the mean-payoff function has a finite number of global maxima around which the behavior of the function is locally Hölder with a known exponent, then the expected regret is bounded up to a logarithmic factor by $\sqrt{n}$, i.e., the rate of the growth of the regret is independent of the dimension of the space. Moreover, we prove the minimax optimality of our algorithm for the class of mean-payoff functions we consider.

## 1 Introduction and motivation

Bandit problems arise in many settings, including clinical trials, scheduling, on-line parameter tuning of algorithms or optimization of controllers based on simulations. In the classical bandit problem there are a finite number of arms that the decision maker can select at discrete time steps. Selecting an arm results in a random reward, whose distribution is determined by the identity of the arm selected. The distributions associated with the arms are unknown to the decision maker whose goal is to maximize the expected sum of the rewards received.

In many practical situations the arms belong to a large set. This set could be continuous [1; 6; 3; 2; 7], hybrid-continuous, or it could be the space of infinite sequences over a finite alphabet [4]. In this paper we consider stochastic bandit problems where the set of arms, $\mathcal{X}$, is allowed to be an arbitrary topological space. We assume that the decision maker knows a dissimilarity function defined over this space that constraints the shape of the mean-payoff function. In particular, the dissimilarity function is assumed to put a lower bound on the mean-payoff function from below at each maxima. We also assume that the decision maker is able to cover the space of arms in a recursive manner, successively refining the regions in the covering such that the diameters of these sets shrink at a known geometric rate when measured with the dissimilarity.

---

[*]Csaba Szepesvári is on leave from MTA SZTAKI. He also greatly acknowledges the support received from the Alberta Ingenuity Fund, iCore and NSERC.

Our work generalizes and improves previous works on continuum-armed bandit problems: Kleinberg [6] and Auer et al. [2] focussed on one-dimensional problems. Recently, Kleinberg et al. [7] considered generic metric spaces assuming that the mean-payoff function is Lipschitz with respect to the (known) metric of the space. They proposed an interesting algorithm that achieves essentially the best possible regret in a minimax sense with respect to these environments.

The goal of this paper is to further these works in a number of ways: *(i)* we allow the set of arms to be a generic topological space; *(ii)* we propose a practical algorithm motivated by the recent very successful tree-based optimization algorithms [8; 5; 4] and show that the algorithm is *(iii)* able to exploit higher order smoothness. In particular, as we shall argue in Section 7, (i) improves upon the results of Auer et al. [2], while (i), (ii) and (iii) improve upon the work of Kleinberg et al. [7]. Compared to Kleinberg et al. [7], our work represents an improvement in the fact that just like Auer et al. [2] we make use of the *local* properties of the mean-payoff function around the maxima only, and not a global property, such as Lipschitzness in the whole space. This allows us to obtain a regret which scales as $\widetilde{O}(\sqrt{n})$ [1] when e.g. the space is the unit hypercube and the mean-payoff function is locally Hölder with known exponent in the neighborhood of any maxima (which are in finite number) and bounded away from the maxima outside of these neighborhoods. Thus, we get the desirable property that the rate of growth of the regret is independent of the dimensionality of the input space. We also prove a minimax lower bound that matches our upper bound up to logarithmic factors, showing that the performance of our algorithm is essentially unimprovable in a minimax sense. Besides these theoretical advances the algorithm is anytime and easy to implement. Since it is based on ideas that have proved to be efficient, we expect it to perform well in practice and to make a significant impact on how on-line global optimization is performed.

## 2   Problem setup, notation

We consider a topological space $\mathcal{X}$, whose elements will be referred to as arms. A decision maker "pulls" the arms in $\mathcal{X}$ one at a time at discrete time steps. Each pull results in a reward that depends on the arm chosen and which the decision maker learns of. The goal of the decision maker is to choose the arms so as to maximize the sum of the rewards that he receives. In this paper we are concerned with stochastic environments. Such an environment $M$ associates to each arm $x \in \mathcal{X}$ a distribution $M_x$ on the real line. The support of these distributions is assumed to be uniformly bounded with a known bound. For the sake of simplicity, we assume this bound is 1. We denote by $f(x)$ the expectation of $M_x$, which is assumed to be measurable (all measurability concepts are with respect to the Borel-algebra over $\mathcal{X}$). The function $f : \mathcal{X} \to \mathbb{R}$ thus defined is called the *mean-payoff function*. When in round $n$ the decision maker pulls arm $X_n \in \mathcal{X}$, he receives a reward $Y_n$ drawn from $M_{X_n}$, independently of the past arm choices and rewards.

A pulling strategy of a decision maker is determined by a sequence $\varphi = (\varphi_n)_{n \geq 1}$ of measurable mappings, where each $\varphi_n$ maps the history space $\mathcal{H}_n = \left( \mathcal{X} \times [0, 1] \right)^{n-1}$ to the space of probability measures over $\mathcal{X}$. By convention, $\varphi_1$ does not take any argument. A strategy is deterministic if for every $n$ the range of $\varphi_n$ contains only Dirac distributions.

According to the process that was already informally described, a pulling strategy $\varphi$ and an environment $M$ jointly determine a random process $(X_1, Y_1, X_2, Y_2, \ldots)$ in the following way: In round one, the decision maker draws an arm $X_1$ at random from $\varphi_1$ and gets a payoff $Y_1$ drawn from $M_{X_1}$. In round $n \geq 2$, first, $X_n$ is drawn at random according to $\varphi_n(X_1, Y_1, \ldots, X_{n-1}, Y_{n-1})$, but otherwise independently of the past. Then the decision maker gets a rewards $Y_n$ drawn from $M_{X_n}$, independently of all other random variables in the past given $X_n$.

Let $f^* = \sup_{x \in \mathcal{X}} f(x)$ be the maximal expected payoff. The *cumulative regret* of a pulling strategy in environment $M$ is $\widehat{R}_n = n f^* - \sum_{t=1}^{n} Y_t$, and the cumulative pseudo-regret is $R_n = n f^* - \sum_{t=1}^{n} f(X_t)$.

In the sequel, we restrict our attention to the expected regret $\mathbb{E}[R_n]$, which in fact equals $\mathbb{E}[\widehat{R}_n]$, as can be seen by the application of the tower rule.

## 3 The Hierarchical Optimistic Optimization (HOO) strategy

### 3.1 Trees of coverings

We first introduce the notion of a tree of coverings. Our algorithm will require such a tree as an input.

**Definition 1** (Tree of coverings). *A tree of coverings is a family of measurable subsets $(\mathcal{P}_{h,i})_{1 \leq i \leq 2^h, \, h \geq 0}$ of $\mathcal{X}$ such that for all fixed integer $h \geq 0$, the covering $\cup_{1 \leq i \leq 2^h} \mathcal{P}_{h,i} = \mathcal{X}$ holds. Moreover, the elements of the covering are obtained recursively: each subset $\mathcal{P}_{h,i}$ is covered by the two subsets $\mathcal{P}_{h+1,2i-1}$ and $\mathcal{P}_{h+1,2i}$.*

A tree of coverings can be represented, as the name suggests, by a binary tree $\mathcal{T}$. The whole domain $\mathcal{X} = \mathcal{P}_{0,1}$ corresponds to the root of the tree and $\mathcal{P}_{h,i}$ corresponds to the $i$–th node of depth $h$, which will be referred to as node $(h,i)$ in the sequel. The fact that each $\mathcal{P}_{h,i}$ is covered by the two subsets $\mathcal{P}_{h+1,2i-1}$ and $\mathcal{P}_{h+1,2i}$ corresponds to the childhood relationship in the tree. Although the definition allows the child-regions of a node to cover a larger part of the space, typically the size of the regions shrinks as depth $h$ increases (cf. Assumption 1).

**Remark 1.** *Our algorithm will instantiate the nodes of the tree on an "as needed" basis, one by one. In fact, at any round $n$ it will only need $n$ nodes connected to the root.*

### 3.2 Statement of the HOO strategy

The algorithm picks at each round a node in the infinite tree $\mathcal{T}$ as follows. In the first round, it chooses the root node $(0,1)$. Now, consider round $n+1$ with $n \geq 1$. Let us denote by $\mathcal{T}_n$ the set of nodes that have been picked in previous rounds and by $\mathcal{S}_n$ the nodes which are not in $\mathcal{T}_n$ but whose parent is. The algorithm picks at round $n+1$ a node $(H_{n+1}, I_{n+1}) \in \mathcal{S}_n$ according to the deterministic rule that will be described below. After selecting the node, the algorithm further chooses an arm $X_{n+1} \in \mathcal{P}_{H_{n+1}, I_{n+1}}$. This selection can be stochastic or deterministic. We do not put any further restriction on it. The algorithm then gets a reward $Y_{n+1}$ as described above and the procedure goes on: $(H_{n+1}, I_{n+1})$ is added to $\mathcal{T}_n$ to form $\mathcal{T}_{n+1}$ and the children of $(H_{n+1}, I_{n+1})$ are added to $\mathcal{S}_n$ to give rise to $\mathcal{S}_{n+1}$. Let us now turn to how $(H_{n+1}, I_{n+1})$ is selected.

Along with the nodes the algorithm stores what we call $B$–values. The node $(H_{n+1}, I_{n+1}) \in \mathcal{S}_n$ to expand at round $n+1$ is picked by following a path from the root to a node in $\mathcal{S}_n$, where at each node along the path the child with the larger $B$–value is selected (ties are broken arbitrarily). In order to define a node's $B$–value, we need a few quantities. Let $\mathcal{C}(h,i)$ be the set that collects $(h,i)$ and its descendants. We let

$$N_{h,i}(n) = \sum_{t=1}^{n} \mathbb{I}_{\{(H_t, I_t) \in \mathcal{C}(h,i)\}}$$

be the number of times the node $(h,i)$ was visited. A given node $(h,i)$ is always picked at most once, but since its descendants may be picked afterwards, subsequent paths in the tree can go through it. Consequently, $1 \leq N_{h,i}(n) \leq n$ for all nodes $(h,i) \in \mathcal{T}_n$. Let $\widehat{\mu}_{h,i}(n)$ be the empirical average of the rewards received for the time-points when the path followed by the algorithm went through $(h,i)$:

$$\widehat{\mu}_{h,i}(n) = \frac{1}{N_{h,i}(n)} \sum_{t=1}^{n} Y_t \, \mathbb{I}_{\{(H_t, I_t) \in \mathcal{C}(h,i)\}}.$$

The corresponding upper confidence bound is by definition

$$U_{h,i}(n) = \widehat{\mu}_{h,i}(n) + \sqrt{\frac{2 \ln n}{N_{h,i}(n)}} + \nu_1 \rho^h,$$

where $0 < \rho < 1$ and $\nu_1 > 0$ are parameters of the algorithm (to be chosen later by the decision maker, see Assumption 1). For nodes not in $\mathcal{T}_n$, by convention, $U_{h,i}(n) = +\infty$. Now, for a node $(h,i)$ in $\mathcal{S}_n$, we define its $B$–value to be $B_{h,i}(n) = +\infty$. The $B$–values for nodes in $\mathcal{T}_n$ are given by

$$B_{h,i}(n) = \min\Big\{U_{h,i}(n), \ \max\big\{B_{h+1,2i-1}(n), \ B_{h+1,2i}(n)\big\}\Big\} .$$

Note that the algorithm is deterministic (apart, maybe, from the arbitrary random choice of $X_t$ in $\mathcal{P}_{H_t,I_t}$). Its total space requirement is linear in $n$ while total running time at round $n$ is at most quadratic in $n$, though we conjecture that it is $O(n \log n)$ on average.

## 4  Assumptions made on the model and statement of the main result

We suppose that $\mathcal{X}$ is equipped with a *dissimilarity* $\ell$, that is a non-negative mapping $\ell : \mathcal{X}^2 \to \mathbb{R}$ satisfying $\ell(x,x) = 0$. The diameter (with respect to $\ell$) of a subset $A$ of $\mathcal{X}$ is given by $\operatorname{diam} A = \sup_{x,y \in A} \ell(x,y)$. Given the dissimilarity $\ell$, the "open" ball with radius $\varepsilon > 0$ and center $c \in \mathcal{X}$ is $\mathcal{B}(c,\varepsilon) = \{ x \in \mathcal{X} : \ell(c,x) < \varepsilon \}$ (we do not require the topology induced by $\ell$ to be related to the topology of $\mathcal{X}$.) In what follows when we refer to an (open) ball, we refer to the ball defined with respect to $\ell$. The dissimilarity will be used to capture the smoothness of the mean-payoff function. The decision maker chooses $\ell$ and the tree of coverings. The following assumption relates this choice to the parameters $\rho$ and $\nu_1$ of the algorithm:

**Assumption 1.** *There exist $\rho < 1$ and $\nu_1, \nu_2 > 0$ such that for all integers $h \geq 0$ and all $i = 1, \ldots, 2^h$, the diameter of $\mathcal{P}_{h,i}$ is bounded by $\nu_1 \rho^h$, and $\mathcal{P}_{h,i}$ contains an open ball $\mathcal{P}'_{h,i}$ of radius $\nu_2 \rho^h$. For a given $h$, the $\mathcal{P}'_{h,i}$ are disjoint for $1 \leq i \leq 2^h$.*

**Remark 2.** *A typical choice for the coverings in a cubic domain is to let the domains be hyper-rectangles. They can be obtained, e.g., in a dyadic manner, by splitting at each step hyper-rectangles in the middle along their longest side, in an axis parallel manner; if all sides are equal, we split them along the first axis. In this example, if $\mathcal{X} = [0,1]^D$ and $\ell(x,y) = \|x - y\|^\alpha$ then we can take $\rho = 2^{-\alpha/D}, \nu_1 = (\sqrt{D}/2)^\alpha$ and $\nu_2 = 1/8^\alpha$.*

The next assumption concerns the environment.

**Definition 2.** *We say that $f$ is* weakly Lipschitz *with respect to $\ell$ if for all $x, y \in \mathcal{X}$,*

$$f^* - f(y) \leq f^* - f(x) + \max\big\{f^* - f(x), \ \ell(x,y)\big\} . \tag{1}$$

Note that weak Lipschitzness is satisfied whenever $f$ is 1–Lipschitz, i.e., for all $x, y \in \mathcal{X}$, one has $|f(x) - f(y)| \leq \ell(x,y)$. On the other hand, weak Lipschitzness implies local (one-sided) 1–Lipschitzness at any maxima. Indeed, at an optimal arm $x^*$ (i.e., such that $f(x^*) = f^*$), (1) rewrites to $f(x^*) - f(y) \leq \ell(x^*,y)$. However, weak Lipschitzness does not constrain the growth of the loss in the vicinity of other points. Further, weak Lipschitzness, unlike Lipschitzness, does not constrain the local *decrease* of the loss at any point. Thus, weak-Lipschitzness is a property that lies somewhere between a growth condition on the loss around optimal arms and (one-sided) Lipschitzness. Note that since weak Lipschitzness is defined with respect to a dissimilarity, it can actually capture higher-order smoothness at the optima. For example, $f(x) = 1 - x^2$ is weak Lipschitz with the dissimilarity $\ell(x,y) = c(x-y)^2$ for some appropriate constant $c$.

**Assumption 2.** *The mean-payoff function $f$ is weakly Lipschitz.*

Let $f^*_{h,i} = \sup_{x \in \mathcal{P}_{h,i}} f(x)$ and $\Delta_{h,i} = f^* - f^*_{h,i}$ be the suboptimality of node $(h,i)$. We say that a node $(h,i)$ is optimal (respectively, suboptimal) if $\Delta_{h,i} = 0$ (respectively, $\Delta_{h,i} > 0$). Let $\mathcal{X}_\varepsilon \stackrel{\text{def}}{=} \{ x \in \mathcal{X} : f(x) \geq f^* - \varepsilon \}$ be the set of $\varepsilon$-optimal arms. The following result follows from the definitions; a proof can be found in the appendix.

**Lemma 1.** *Let Assumption 1 and 2 hold. If the suboptimality $\Delta_{h,i}$ of a region is bounded by $c\nu_1\rho^h$ for some $c > 0$, then all arms in $\mathcal{P}_{h,i}$ are $\max\{2c, c+1\}\nu_1\rho^h$-optimal.*

The last assumption is closely related to Assumption 2 of Auer et al. [2], who observed that the regret of a continuum-armed bandit algorithm should depend on how fast the volume of the sets of $\varepsilon$-optimal arms shrinks as $\varepsilon \to 0$. Here, we capture this by defining a new notion, the near-optimality dimension of the mean-payoff function. The connection between these concepts, as well as the zooming dimension defined by Kleinberg et al. [7] will be further discussed in Section 7.

Define the packing number $\mathcal{P}(\mathcal{X}, \ell, \varepsilon)$ to be the size of the largest packing of $\mathcal{X}$ with disjoint open balls of radius $\varepsilon$ with respect to the dissimilarity $\ell$.[2] We now define the near-optimality dimension, which characterizes the size of the sets $\mathcal{X}_\varepsilon$ in terms of $\varepsilon$, and then state our main result.

**Definition 3.** *For $c > 0$ and $\varepsilon_0 > 0$, the $(c, \varepsilon_0)$–near-optimality dimension of $f$ with respect to $\ell$ equals*

$$\inf\Big\{d \in [0, +\infty) \ : \ \exists C \text{ s.t. } \forall \varepsilon \leq \varepsilon_0, \ \ \mathcal{P}\big(\mathcal{X}_{c\varepsilon}, \ell, \varepsilon\big) \leq C\,\varepsilon^{-d}\Big\} \tag{2}$$

*(with the usual convention that $\inf \emptyset = +\infty$).*

**Theorem 1** (Main result). *Let Assumptions 1 and 2 hold and assume that the $(4\nu_1/\nu_2, \nu_2)$–near-optimality dimension of the considered environment is $d < +\infty$. Then, for any $d' > d$ there exists a constant $C(d')$ such that for all $n \geq 1$,*

$$\mathbb{E}R_n \leq C(d')\,n^{(d'+1)/(d'+2)}\left(\ln n\right)^{1/(d'+2)}.$$

*Further, if the near-optimality dimension is achieved, i.e., the infimum is achieved in (2), then the result holds also for $d' = d$.*

**Remark 3.** *We can relax the weak-Lipschitz property by requiring it to hold only locally around the maxima. In fact, at the price of increased constants, the result continues to hold if there exists $\varepsilon > 0$ such that (1) holds for any $x, y \in \mathcal{X}_\varepsilon$. To show this we only need to carefully adapt the steps of the proof below. We omit the details from this extended abstract.*

## 5  Analysis of the regret and proof of the main result

We first state three lemmas, whose proofs can be found in the appendix. The proofs of Lemmas 3 and 4 rely on concentration-of-measure techniques, while that of Lemma 2 follows from a simple case study. Let us fix some path $(0, 1), (1, i_1^*), \dots, (h, i_h^*), \dots,$ of optimal nodes, starting from the root.

**Lemma 2.** *Let $(h, i)$ be a suboptimal node. Let $k$ be the largest depth such that $(k, i_k^*)$ is on the path from the root to $(h, i)$. Then we have*

$$\mathbb{E}\big[N_{h,i}(n)\big] \leq u + \sum_{t=u+1}^{n} \mathbb{P}\Big\{N_{h,i}(t) > u \text{ and } \big[U_{h,i}(t) > f^* \text{ or } U_{s,i_s^*} \leq f^* \text{ for some } s \in \{k+1, \dots, t-1\}\big]\Big\}.$$

**Lemma 3.** *Let Assumptions 1 and 2 hold. Then, for all optimal nodes and for all integers $n \geq 1$, $\mathbb{P}\big\{U_{h,i}(n) \leq f^*\big\} \leq n^{-3}$.*

**Lemma 4.** *Let Assumptions 1 and 2 hold. Then, for all integers $t \leq n$, for all suboptimal nodes $(h, i)$ such that $\Delta_{h,i} > \nu_1\rho^h$, and for all integers $u \geq 1$ such that $u \geq \frac{8\ln n}{(\Delta_{h,i} - \nu_1\rho^h)^2}$, one has $\mathbb{P}\big\{U_{h,i}(t) > f^* \text{ and } N_{h,i}(t) > u\big\} \leq t\,n^{-4}$.*

Taking $u$ as the integer part of $(8 \ln n)/(\Delta_{h,i} - \nu_1 \rho^h)^2$, and combining the results of Lemma 2, 3, and 4 with a union bound leads to the following key result.

**Lemma 5.** *Under Assumptions 1 and 2, for all suboptimal nodes $(h,i)$ such that $\Delta_{h,i} > \nu_1 \rho^h$, we have, for all $n \geq 1$,*

$$\mathbb{E}[N_{h,i}(n)] \leq \frac{8 \ln n}{(\Delta_{h,i} - \nu_1 \rho^h)^2} + \frac{2}{n} \ .$$

We are now ready to prove Theorem 1.

*Proof.* For the sake of simplicity we assume that the infimum in the definition of near-optimality is achieved. To obtain the result in the general case one only needs to replace $d$ below by $d' > d$ in the proof below.

**First step.** For all $h = 1, 2, \ldots$, denote by $\mathcal{I}_h$ the nodes at depth $h$ that are $2\nu_1 \rho^h$–optimal, i.e., the nodes $(h,i)$ such that $f^*_{h,i} \geq f^* - 2\nu_1 \rho^h$. Then, $\mathcal{I}$ is the union of these sets of nodes. Further, let $\mathcal{J}$ be the set of nodes that are not in $\mathcal{I}$ but whose parent is in $\mathcal{I}$. We then denote by $\mathcal{J}_h$ the nodes in $\mathcal{J}$ that are located at depth $h$ in the tree. Lemma 4 bounds the expected number of times each node $(h,i) \in \mathcal{J}_h$ is visited. Since $\Delta_{h,i} > 2\nu_1 \rho^h$, we get

$$\mathbb{E}[N_{h,i}(n)] \leq \frac{8 \ln n}{\nu_1^2 \rho^{2h}} + \frac{2}{n} \ .$$

**Second step.** We bound here the cardinality $|\mathcal{I}_h|$, $h > 0$. If $(h,i) \in \mathcal{I}_h$ then since $\Delta_{h,i} \leq 2\nu_1 \rho^h$, by Lemma 1 $\mathcal{P}_{h,i} \subset \mathcal{X}_{4\nu_1\rho^h}$. Since by Assumption 1, the sets $(\mathcal{P}_{h,i})$, for $(h,i) \in \mathcal{I}_h$, contain disjoint balls of radius $\nu_2 \rho^h$, we have that

$$|\mathcal{I}_h| \leq \mathcal{P}\big(\cup_{(h,i)\in\mathcal{I}_h}\mathcal{P}_{h,i},\ \ell,\ \nu_2\rho^h\big) \leq \mathcal{P}\big(\mathcal{X}_{(4\nu_1/\nu_2)\,\nu_2\rho^h},\ \ell,\ \nu_2\rho^h\big) \leq C\left(\nu_2\rho^h\right)^{-d} ,$$

where we used the assumption that $d$ is the $(4\nu_1/\nu_2, \nu_2)$–near-optimality dimension of $f$ (and $C$ is the constant introduced in the definition of the near-optimality dimension).

**Third step.** Choose $\eta > 0$ and let $H$ be the smallest integer such that $\rho^H \leq \eta$. We partition the infinite tree $\mathcal{T}$ into three sets of nodes, $\mathcal{T} = \mathcal{T}_1 \cup \mathcal{T}_2 \cup \mathcal{T}_3$. The set $\mathcal{T}_1$ contains nodes of $\mathcal{I}_H$ and their descendants, $\mathcal{T}_2 = \cup_{0 \leq h < H} \mathcal{I}_h$, and $\mathcal{T}_3$ contains the nodes $\cup_{1 \leq h \leq H} \mathcal{J}_h$ and their descendants. (Note that $\mathcal{T}_1$ and $\mathcal{T}_3$ are potentially infinite, while $\mathcal{T}_2$ is finite.)

We denote by $(H_t, I_t)$ the node that was chosen by the forecaster at round $t$ to pick $X_t$. From the definition of the forecaster, no two such random variables are equal, since each node is picked at most once. We decompose the regret according to the element $\mathcal{T}_j$ where the chosen nodes $(H_t, I_t)$ belong to:

$$\mathbb{E}[R_n] = \mathbb{E}\left[\sum_{t=1}^n (f^* - f(X_t))\right] = \mathbb{E}[R_{n,1}] + \mathbb{E}[R_{n,2}] + \mathbb{E}[R_{n,3}],$$

$$\text{where for all } i = 1,\,2,\,3, \qquad R_{n,i} = \sum_{t=1}^n (f^* - f(X_t))\mathbb{I}_{\{(H_t,I_t)\in\mathcal{T}_i\}} \ .$$

The contribution from $\mathcal{T}_1$ is easy to bound. By definition any node in $\mathcal{I}_H$ is $2\nu_1\rho^H$-optimal. Hence, by Lemma 1 the corresponding domain is included in $\mathcal{X}_{4\nu_1\rho^H}$. The domains of these nodes' descendants are of course still included in $\mathcal{X}_{4\nu_1\rho^H}$. Therefore, $\mathbb{E}[R_{n,1}] \leq 4n\nu_1\rho^H$.

For $h \geq 1$, consider a node $(h,i) \in \mathcal{T}_2$. It belongs to $\mathcal{I}_h$ and is therefore $2\nu_1\rho^h$–optimal. By Lemma 1, the corresponding domain is included in $\mathcal{X}_{4\nu_1\rho^h}$. By the result of the second step and using that each node is played at most once, one gets

$$\mathbb{E}[R_{n,2}] \leq \sum_{h=0}^{H-1} 4\nu_1\rho^h \,|\mathcal{I}_h| \leq 4\nu_1 C \,\nu_2^{-d} \sum_{h=0}^{H-1} \rho^{h(1-d)} \ .$$

We finish with the contribution from $\mathcal{T}_3$. We first remark that since the parent of any element $(h, i) \in \mathcal{J}_h$ is in $\mathcal{I}_{h-1}$, by Lemma 1 again, we have that $\mathcal{P}_{h,i} \subset \mathcal{X}_{4\nu_1 \rho^{h-1}}$. To each node $(H_t, I_t)$ played in $\mathcal{T}_3$, we associate the element $(H'_t, I'_t)$ of some $\mathcal{J}_h$ on the path from the root to $(H_t, I_t)$. When $(H_t, I_t)$ is played, the chosen arm $X_t$ belongs also to $\mathcal{P}_{H'_t, I'_t}$. Decomposing $R_{n,3}$ according to the elements of $\cup_{1 \le h \le H} \mathcal{J}_h$, we then bound the regret from $\mathcal{T}_3$ as

$$\mathbb{E}\big[R_{n,3}\big] \le \sum_{h=1}^{H} 4\nu_1 \rho^{h-1} \sum_{i \,:\, (h,i)\in\mathcal{J}_h} \mathbb{E}\big[N_{h,i}(n)\big] \le \sum_{h=1}^{H} 4\nu_1 \rho^{h-1} |\mathcal{J}_h| \left( \frac{8\ln n}{\nu_1^2 \rho^{2h}} + \frac{2}{n} \right)$$

where we used the result of the first step. Now, it follows from that fact that the parent of $\mathcal{J}_h$ is in $\mathcal{I}_{h-1}$ that $|\mathcal{J}_h| \le 2|\mathcal{I}_{h-1}|$. Substituting this and the bound on $|\mathcal{I}_{h-1}|$, we get

$$\mathbb{E}\big[R_{n,3}\big] \le 8\nu_1 C \nu_2^{-d} \sum_{h=1}^{H} \rho^{h(1-d)+d-1} \left( \frac{8\ln n}{\nu_1^2 \rho^{2h}} + \frac{2}{n} \right) .$$

**Fourth step.** Putting things together, we have proved

$$
\begin{aligned}
\mathbb{E}\big[R_n\big] &\le 4n\nu_1 \rho^H + 4\nu_1 C \nu_2^{-d} \sum_{h=0}^{H-1} \rho^{h(1-d)} + 8\nu_1 C \nu_2^{-d} \sum_{h=1}^{H} \rho^{h(1-d)+d-1} \left( \frac{8\ln n}{\nu_1^2 \rho^{2h}} + \frac{2}{n} \right) \\
&= O\left( n\rho^H + (\ln n) \sum_{h=1}^{H} \rho^{-h(1+d)} \right) = O\left( n\rho^H + \rho^{-H(1+d)} \ln n \right) = O\left( n^{(d+1)/(d+2)} (\ln n)^{1/(d+2)} \right)
\end{aligned}
$$

by using first that $\rho < 1$ and then, by optimizing over $\rho^H$ (the worst value being $\rho^H \sim \left(\frac{n}{\ln n}\right)^{-1/(d+2)}$). $\quad\square$

# 6 Minimax optimality

The packing dimension of a set $\mathcal{X}$ is the smallest $d$ such that there exists a constant $k$ such that for all $\varepsilon > 0$, $\mathcal{P}(\mathcal{X}, \ell, \varepsilon) \le k\,\varepsilon^{-d}$. For instance, compact subsets of $\mathbb{R}^d$ (with non-empty interior) have a packing dimension of $d$ whenever $\ell$ is a norm. If $\mathcal{X}$ has a packing dimension of $d$, then all environments have a near-optimality dimension less than $d$. The proof of the main theorem indicates that the constant $C(d)$ only depends on $d$, $k$ (of the definition of packing dimension), $\nu_1$, $\nu_2$, and $\rho$, but not on the environment as long as it is weakly Lipschitz. Hence, we can extract from it a distribution-free bound of the form $\widetilde{O}(n^{(d+1)/(d+2)})$. In fact, this bound can be shown to be optimal as is illustrated by the theorem below, whose assumptions are satisfied by, e.g., compact subsets of $\mathbb{R}^d$ and if $\ell$ is some norm of $\mathbb{R}^d$. The proof can be found in the appendix.

**Theorem 2.** *If $\mathcal{X}$ is such that there exists $c > 0$ with $\mathcal{P}(\mathcal{X}, \ell, \varepsilon) \ge c\,\varepsilon^{-d} \ge 2$ for all $\varepsilon \le 1/4$ then for all $n \ge 4^{d-1} c/\ln(4/3)$, all strategies $\varphi$ are bound to suffer a regret of at least*

$$\sup \mathbb{E}\, R_n(\varphi) \ge \frac{1}{4} \left( \frac{1}{4} \sqrt{\frac{c}{4\ln(4/3)}} \right)^{2/(d+2)} n^{(d+1)/(d+2)},$$

*where the supremum is taken over all environments with weakly Lipschitz payoff functions.*

# 7 Discussion

Several works [1; 6; 3; 2; 7] have considered continuum-armed bandits in Euclidean or metric spaces and provided upper- and lower-bounds on the regret for given classes of environments. Cope [3] derived a regret of $\widetilde{O}(\sqrt{n})$ for compact and convex subset of $R^d$ and a mean-payoff function with unique minima and second order smoothness. Kleinberg [6] considered mean-payoff functions $f$ on the real line that are Hölder with degree $0 < \alpha \le 1$. The derived regret is $\Theta(n^{(\alpha+1)/(\alpha+2)})$. Auer et al. [2] extended the analysis to classes of functions with only a local Hölder assumption around maximum (with possibly higher smoothness degree $\alpha \in [0, \infty)$), and derived the regret $\Theta(n^{\frac{1+\alpha-\alpha\beta}{1+2\alpha-\alpha\beta}})$, where $\beta$ is such that the Lebesgue measure of $\varepsilon$-optimal

states is $O(\varepsilon^\beta)$. Another setting is that of [7] who considered a metric space $(\mathcal{X}, \ell)$ and assumed that $f$ is Lipschitz w.r.t. $\ell$. The obtained regret is $\widetilde{O}(n^{(d+1)/(d+2)})$ where $d$ is the zooming dimension (defined similarly to our near-optimality dimension, but using covering numbers instead of packing numbers and the sets $\mathcal{X}_\varepsilon \setminus \mathcal{X}_{\varepsilon/2}$). When $(\mathcal{X}, \ell)$ is a metric space covering and packing numbers are equivalent and we may prove that the zooming dimension and near-optimality dimensions are equal.

Our main contribution compared to [7] is that our weak-Lipschitz assumption, which is substantially weaker than the global Lipschitz assumption assumed in [7], enables our algorithm to work better in some common situations, such as when the mean-payoff function assumes a local smoothness whose order is larger than one. In order to relate all these results, let us consider a specific example: Let $\mathcal{X} = [0,1]^D$ and assume that the mean-reward function $f$ is locally equivalent to a Hölder function with degree $\alpha \in [0, \infty)$ around any maxima $x^*$ of $f$ (the number of maxima is assumed to be finite):

$$f(x^*) - f(x) = \Theta(||x - x^*||^\alpha) \text{ as } x \to x^*. \tag{3}$$

This means that $\exists c_1, c_2, \varepsilon_0 > 0$, $\forall x$, s.t. $||x - x^*|| \leq \varepsilon_0$, $c_1||x - x^*||^\alpha \leq f(x^*) - f(x) \leq c_2||x - x^*||^\alpha$. Under this assumption, the result of Auer et al. [2] shows that for $D = 1$, the regret is $\Theta(\sqrt{n})$ (since here $\beta = 1/\alpha$). Our result allows us to extend the $\sqrt{n}$ regret rate to any dimension $D$. Indeed, if we choose our dissimilarity measure to be $\ell_\alpha(x, y) \overset{\text{def}}{=} ||x - y||^\alpha$, we may prove that $f$ satisfies a locally weak-Lipschitz condition (as defined in Remark 3) and that the near-optimality dimension is $0$. Thus our regret is $\widetilde{O}(\sqrt{n})$, i.e., the rate is independent of the dimension $D$.

In comparison, since Kleinberg et al. [7] have to satisfy a global Lipschitz assumption, they can not use $\ell_\alpha$ when $\alpha > 1$. Indeed a function globally Lipschitz with respect to $\ell_\alpha$ is essentially constant. Moreover $\ell_\alpha$ does not define a metric for $\alpha > 1$. If one resort to the Euclidean metric to fulfill their requirement that $f$ be Lipschitz w.r.t. the metric then the zooming dimension becomes $D(\alpha - 1)/\alpha$, while the regret becomes $\widetilde{O}(n^{(D(\alpha-1)+\alpha)/(D(\alpha-1)+2\alpha)})$, which is strictly worse than $\widetilde{O}(\sqrt{n})$ and in fact becomes close to the slow rate $\widetilde{O}(n^{(D+1)/(D+2)})$ when $\alpha$ is larger. Nevertheless, in the case of $\alpha \leq 1$ they get the same regret rate.

In contrast, our result shows that under very weak constraints on the mean-payoff function and if the local behavior of the function around its maximum (or finite number of maxima) is known then global optimization suffers a regret of order $\widetilde{O}(\sqrt{n})$, independent of the space dimension. As an interesting sidenote let us also remark that our results allow different smoothness orders along different dimensions, i.e., heterogenous smoothness spaces.

## Footnotes

[1] We write $u_n = \widetilde{O}(v_u)$ when $u_n = O(v_n)$ up to a logarithmic factor.

[2] Note that sometimes packing numbers are defined as the largest packing with disjoint open balls of radius $\varepsilon/2$, or, $\varepsilon$-nets.

## References

[1] R. Agrawal. The continuum-armed bandit problem. *SIAM J. Control and Optimization*, 33:1926–1951, 1995.

[2] P. Auer, R. Ortner, and Cs. Szepesvári. Improved rates for the stochastic continuum-armed bandit problem. *20th Conference on Learning Theory*, pages 454–468, 2007.

[3] E. Cope. Regret and convergence bounds for immediate-reward reinforcement learning with continuous action spaces. Preprint, 2004.

[4] P.-A. Coquelin and R. Munos. Bandit algorithms for tree search. In *Proceedings of 23rd Conference on Uncertainty in Artificial Intelligence*, 2007.

[5] S. Gelly, Y. Wang, R. Munos, and O. Teytaud. Modification of UCT with patterns in Monte-Carlo go. Technical Report RR-6062, INRIA, 2006.

[6] R. Kleinberg. Nearly tight bounds for the continuum-armed bandit problem. In *18th Advances in Neural Information Processing Systems*, 2004.

[7] R. Kleinberg, A. Slivkins, and E. Upfal. Multi-armed bandits in metric spaces. In *Proceedings of the 40th ACM Symposium on Theory of Computing*, 2008.

[8] L. Kocsis and Cs. Szepesvári. Bandit based Monte-Carlo planning. In *Proceedings of the 15th European Conference on Machine Learning*, pages 282–293, 2006.

